# Variance Reduction Techniques for Gradient Estimates in Reinforcement Learning

**Evan Greensmith**
Australian National University
evan@csl.anu.edu.au

**Peter L. Bartlett**[*]
BIOwulf Technologies
Peter.Bartlett@anu.edu.au

**Jonathan Baxter**[*]
WhizBang! Labs, East
jbaxter@whizbang.com

## Abstract

We consider the use of two additive control variate methods to reduce the variance of performance gradient estimates in reinforcement learning problems. The first approach we consider is the baseline method, in which a function of the current state is added to the discounted value estimate. We relate the performance of these methods, which use sample paths, to the variance of estimates based on iid data. We derive the baseline function that minimizes this variance, and we show that the variance for any baseline is the sum of the optimal variance and a weighted squared distance to the optimal baseline. We show that the widely used average discounted value baseline (where the reward is replaced by the difference between the reward and its expectation) is suboptimal. The second approach we consider is the actor-critic method, which uses an approximate value function. We give bounds on the expected squared error of its estimates. We show that minimizing distance to the true value function is suboptimal in general; we provide an example for which the true value function gives an estimate with positive variance, but the optimal value function gives an unbiased estimate with zero variance. Our bounds suggest algorithms to estimate the gradient of the performance of parameterized baseline or value functions. We present preliminary experiments that illustrate the performance improvements on a simple control problem.

## 1 Introduction, Background, and Preliminary Results

In reinforcement learning problems, the aim is to select a controller that will maximize the average reward in some environment. We model the environment as a partially observable Markov decision process (POMDP). Gradient ascent methods (e.g., [7, 12, 15]) estimate the gradient of the average reward, usually using Monte Carlo techniques to cal-

---

[*]Most of this work was performed while the authors were with the Research School of Information Sciences and Engineering at the Australian National University.

culate an average over a sample path of the controlled POMDP. However such estimates tend to have a high variance, which means many steps are needed to obtain a good estimate. GPOMDP [4] is an algorithm for generating an estimate of the gradient in this way. Compared with other approaches, it is suitable for large systems, when the time between visits to a state is large but the mixing time of the controlled POMDP is short. However, it can suffer from the problem of producing high variance estimates. In this paper, we investigate techniques for variance reduction in GPOMDP. One generic approach to reducing the variance of Monte Carlo estimates of integrals is to use an additive control variate (see, for example, [6]). Suppose we wish to estimate the integral of $f : \mathcal{X} \to \mathbb{R}$, and we know the integral of another function $\varphi : \mathcal{X} \to \mathbb{R}$. Since $\int_{\mathcal{X}} f = \int_{\mathcal{X}} (f - \varphi) + \int_{\mathcal{X}} \varphi$, the integral of $f - \varphi$ can be estimated instead. Obviously if $\varphi = f$ then the variance is zero. More generally, $\mathrm{Var}(f - \varphi) = \mathrm{Var}(f) - 2\mathrm{Cov}(f, \varphi) + \mathrm{Var}(\varphi)$, so that if $\phi$ and $f$ are strongly correlated, the variance of the estimate is reduced.

In this paper, we consider two approaches of this form. The first (Section 2) is the technique of adding a baseline. We find the optimal baseline and we show that the additional variance of a suboptimal baseline can be expressed as a weighted squared distance from the optimal baseline. Constant baselines, which do not depend on the state or observations, have been widely used [13, 15, 9, 11]. In particular, the expectation over all states of the discounted value of the state is a popular constant baseline (where, for example, the reward at each step is replaced by the difference between the reward and the expected reward). We give bounds on the estimation variance that show that, perhaps surprisingly, this may not be the best choice.

The second approach (Section 3) is the use of an approximate value function. Such *actor-critic methods* have been investigated extensively [3, 1, 14, 10]. Generally the idea is to minimize some notion of distance between the fixed value function and the true value function. In this paper we show that this may not be the best approach: selecting the fixed value function to be equal to the true value function is not always the best choice. Even more surprisingly, we give an example for which the use of a fixed value function that is different from the true value function reduces the variance to zero, for no increase in bias. We give a bound on the expected squared error (that is, including the estimation variance) of the gradient estimate produced with a fixed value function. Our results suggest new algorithms to learn the optimum baseline, and to learn a fixed value function that minimizes the bound on the error of the estimate. In Section 5, we describe the results of preliminary experiments, which show that these algorithms give performance improvements.

## POMDP with Reactive, Parameterized Policy

A partially observable Markov decision process (POMDP) consists of a state space, $\mathcal{S}$, a control space, $\mathcal{U}$, an observation space, $\mathcal{Y}$, a set of transition probability matrices $\{P(u) : u \in \mathcal{U}\}$, each with components $\mathrm{p}_{ij}(u)$ for $i, j \in \mathcal{S}, u \in \mathcal{U}$, an observation process $\nu : \mathcal{S} \to \mathcal{P}_{\mathcal{Y}}$, where $\mathcal{P}_{\mathcal{Y}}$ is the space of probability distributions over $\mathcal{Y}$, and a reward function $\mathrm{r} : \mathcal{S} \to \mathbb{R}$. We assume that $\mathcal{S}, \mathcal{U}, \mathcal{Y}$ are finite, although all our results extend easily to infinite $\mathcal{U}$ and $\mathcal{Y}$, and with more restrictive assumptions can be extended to infinite $\mathcal{S}$. A reactive, parameterized policy for a POMDP is a set of mappings $\{\mu(\cdot, \theta) : \mathcal{Y} \to \mathcal{P}_{\mathcal{U}} | \theta \in \mathbb{R}^K\}$. Together with the POMDP, this defines the *controlled POMDP* $(\mathcal{S}, \mathcal{U}, \mathcal{Y}, P, \nu, \mathrm{r}, \mu)$. The joint state, observation and control process, $\{X_t, Y_t, U_t\}$, is Markov. The state process, $\{X_t\}$, is also Markov, with transition probabilities $\mathrm{p}_{ij}(\theta) = \sum_{y \in \mathcal{Y}, u \in \mathcal{U}} \nu_y(i) \mu_u(y, \theta) \mathrm{p}_{ij}(u)$, where $\nu_y(i)$ denotes the probability of observation $y$ given the state $i$, and $\mu_u(y, \theta)$ denotes the probability of action $u$ given parameters $\theta$ and observation $y$. The Markov chain $\mathrm{M}(\theta) = (\mathcal{S}, P(\theta))$ then describes the behaviour of the process $\{X_t\}$.

**Assumption 1** *The controlled POMDP* $(\mathcal{S}, \mathcal{U}, \mathcal{Y}, P, \nu, r, \mu)$ *satisfies:*
*For all* $\theta \in \mathbb{R}^K$ *there exists a unique stationary distribution satisfying* $\pi'(\theta) P(\theta) = \pi'(\theta)$.
*There is an* $\mathbf{R} < \infty$ *such that, for all* $i \in \mathcal{S}$, $|r(i)| \leq \mathbf{R}$.
*There is a* $\mathbf{B} < \infty$ *such that, for all* $u \in \mathcal{U}$, $y \in \mathcal{Y}$ *and* $\theta \in \mathbb{R}^K$ *the derivatives* $\partial \mu_u(y, \theta) / \partial \theta_k$ $(1 \leq k \leq K)$ *exist, and the vector of these derivatives satisfies* $\|\nabla \mu_u(y, \theta) / \mu_u(y, \theta)\| \leq \mathbf{B}$, *where* $\| \cdot \|$ *denotes the Euclidean norm on* $\mathbb{R}^K$.

We consider the average reward, $\eta(\theta) \stackrel{\text{def}}{=} \lim_{T \to \infty} \mathbb{E}\left[\frac{1}{T} \sum_{t=0}^{T-1} r(X_t)\right]$. Assumption 1 implies that this limit exists, and does not depend on the start state $X_0$. The aim is to select a policy to maximize this quantity. Define the discounted value function, $J_\beta(i, \theta) \stackrel{\text{def}}{=} \lim_{T \to \infty} \mathbb{E}\left[\sum_{t=0}^{T-1} \beta^t r(X_t) \Big| X_0 = i\right]$. Throughout the rest of the paper, dependences upon $\theta$ are assumed, and dropped in the notation. For a random vector $A$, we denote $\text{Var}(A) = \mathbb{E}\left[(A - \mathbb{E}[A])^2\right]$, where $a^2$ denotes $a'a$, and $a'$ denotes the transpose of the column vector $a$.

## GPOMDP Algorithm

The GPOMDP algorithm [4] uses a sample path to estimate the gradient approximation $\nabla_\beta \eta \stackrel{\text{def}}{=} \mathbb{E}\left[\frac{\nabla \mu_u(y)}{\mu_u(y)} J_\beta(j)\right]$. As $\beta \to 1$, $\nabla_\beta \eta$ approaches the true gradient $\nabla \eta$, but the variance increases. We consider a slight modification [2]: with $J_t \stackrel{\text{def}}{=} \sum_{s=t}^{2T} \beta^{s-t} r(X_s)$,

$$\Delta_T \stackrel{\text{def}}{=} \frac{1}{T} \sum_{t=0}^{T-1} \frac{\nabla \mu_{U_t}(Y_t)}{\mu_{U_t}(Y_t)} J_{t+1}. \tag{1}$$

Throughout this paper the process $\{X_t, Y_t, U_t, X_{t+1}\}$ is generally understood to be generated by a controlled POMDP satisfying Assumption 1, with $X_0 \sim \pi$ (ie the initial state distributed according to the stationary distribution). That is, before computing the gradient estimates, we wait until the process has settled down to the stationary distribution.

## Dependent Samples

Correlation terms arise in the variance quantities to be analysed. We show here that considering iid samples gives an upper bound on the variance of the general case. The mixing time of a finite ergodic Markov chain $M = (\mathcal{S}, P)$ is defined as

$$\tau \stackrel{\text{def}}{=} \min\left\{t > 1 : \max_{i,j} d_{TV}\left([P^t]_i, [P^t]_j\right) \leq e^{-1}\right\},$$

where $[P^t]_i$ denotes the $i$th row of $P^t$ and $d_{TV}$ is the total variation distance, $d_{TV}(P, Q) = \sum_i |P(i) - Q(i)|$.

**Theorem 1** *Let* $M = (\mathcal{S}, P)$ *be a finite ergodic Markov chain, with mixing time* $\tau$, *and let* $\pi$ *be its stationary distribution. There are constants* $\mathbf{L} < \sqrt{2|\mathcal{S}|e}$ *and* $0 \leq \alpha < \exp(-1/(2\tau))$, *which depend only on* $M$, *such that, for all* $f : \mathcal{S} \to \mathbb{R}$ *and all* $t$, $\left|\text{Cov}_f^\pi(t)\right| \leq \mathbf{L}\alpha^t \text{Var}_\pi(f)$, *where* $\text{Var}_\pi(f)$ *is the variance of* $f$ *under* $\pi$, *and* $\text{Cov}_f^\pi(t)$ *is the auto-covariance of the process* $\{f(X_t)\}$, *where the process* $\{X_t\}$ *is generated by* $M$ *with initial distribution* $\pi$. *Hence, for some constant* $\Omega^* \leq 4\mathbf{L}\tau$,

$$\text{Var}\left(\frac{1}{T} \sum_{t=0}^{T-1} f(X_t)\right) \leq \frac{\Omega^*}{T} \text{Var}_\pi(f).$$

We call $(\mathbf{L}, \tau)$ the *mixing constants* of the Markov chain $M$ (or of the controlled POMDP $D$; ie the Markov chain $(\mathcal{S}, P)$). We omit the proof (all proofs are in the full version [8]). Briefly, we show that for a finite ergodic Markov chain $M$, $\left|\mathrm{Cov}_{\mathrm{f}}^{\pi}(t)\right| \leq \mathrm{R}_t(M) \mathrm{Var}_\pi(\mathrm{f})$, for some $\mathrm{R}_t(M)$. We then show that $\mathrm{R}_t(M)^2 < 2|\mathcal{S}| \exp(-\lfloor \frac{t}{\tau} \rfloor)$. In fact, for a reversible chain, we can choose $\mathbf{L} = 1$ and $\alpha = |\lambda_2|$, the second largest magnitude eigenvalue of $P$.

## 2   Baseline

We consider an alteration of (1),

$$
\underline{\Delta}_T \stackrel{\text{def}}{=} \frac{1}{T} \sum_{t=0}^{T-1} \frac{\nabla \mu_{U_t}(Y_t)}{\mu_{U_t}(Y_t)} \left( \mathrm{J}_{t+1} - \mathrm{A}_r(Y_t) \right). \tag{2}
$$

For any baseline $\mathrm{A}_r : \mathcal{Y} \to \mathbb{R}$, it is easy to show that $\mathbb{E}\left[\underline{\Delta}_T\right] = \mathbb{E}\left[\Delta_T\right]$. Thus, we select $\mathrm{A}_r$ to minimize variance. The following theorem shows that this variance is bounded by a variance involving iid samples, with $\mathrm{J}_t$ replaced by the exact value function.

**Theorem 2** *Suppose that $D = (\mathcal{S}, \mathcal{U}, \mathcal{Y}, P, \nu, \mathrm{r}, \mu)$ is a controlled POMDP satisfying Assumption 1, $D$ has mixing constants $(\mathbf{L}, \tau)$, $\{X_t, Y_t, U_t, X_{t+1}\}$ is a process generated by $D$ with $X_0 \sim \pi$, $\mathrm{A}_r : \mathcal{Y} \to \mathbb{R}$ is a baseline that is uniformly bounded by $\mathbf{M}$, and $\mathcal{J}(j)$ has the distribution of $\sum_{s=0}^{\infty} \beta^s r(X_t)$, where the states $X_t$ are generated by $D$ starting in $X_0 = j$. Then there are constants $C \leq 5\mathbf{B}^2 \mathbf{R}(\mathbf{R} + \mathbf{M})$ and $\Omega \leq 4\mathbf{L}\tau \ln(eT)$ such that*

$$
\mathrm{Var}\left( \frac{1}{T} \sum_{t=0}^{T-1} \frac{\nabla \mu_{U_t}(Y_t)}{\mu_{U_t}(Y_t)} \left( \mathrm{J}_{t+1} - \mathrm{A}_r(Y_t) \right) \right) \leq \frac{\Omega}{T} \mathrm{Var}_\pi\left( \frac{\nabla \mu_u(y)}{\mu_u(y)} \left( \mathrm{J}_\beta(j) - \mathrm{A}_r(y) \right) \right)
$$

$$
+ \frac{\Omega}{T} \mathbb{E}\left( \frac{\nabla \mu_u(y)}{\mu_u(y)} \left( \mathcal{J}(j) - \mathrm{J}_\beta(j) \right) \right)^2 + \left( \frac{\Omega}{T} + 1 \right) \frac{C}{(1 - \beta)^2} \beta^T,
$$

*where, as always, $(i, y, u, j)$ are generated iid with $i \sim \pi$, $y \sim \nu(i)$, $u \sim \mu(y)$ and $j \sim P_i(u)$.*

The proof uses Theorem 1 and [2, Lemma 4.3]. Here we have bounded the variance of (2) with the variance of a quantity we may readily analyse. The second term on the right hand side shows the error associated with replacing an unbiased, uncorrelated estimate of the value function with the true value function. This quantity is not dependent on the baseline. The final term on the right hand side arises from the truncation of the discounted reward— and is exponentially decreasing. We now concentrate on minimizing the variance

$$
\underline{\sigma}_r^2 = \mathrm{Var}_\pi\left( \frac{\nabla \mu_u(y)}{\mu_u(y)} \left( \mathrm{J}_\beta(j) - \mathrm{A}_r(y) \right) \right), \tag{3}
$$

which the following lemma relates to the variance $\sigma^2$ without a baseline,

$$
\sigma^2 = \mathrm{Var}_\pi\left( \frac{\nabla \mu_u(y)}{\mu_u(y)} \mathrm{J}_\beta(j) \right).
$$

**Lemma 3** *Let $D = (\mathcal{S}, \mathcal{U}, \mathcal{Y}, P, \nu, \mathrm{r}, \mu)$ be a controlled POMDP satisfying Assumption 1. For any baseline $\mathrm{A}_r : \mathcal{Y} \to \mathbb{R}$, and for $i \sim \pi$, $y \sim \nu(i)$, $u \sim \mu(y)$ and $j \sim P_i(u)$,*

$$
\underline{\sigma}_r^2 = \sigma^2 + \mathbb{E}\left[ \mathrm{A}_r^2(y)\, \mathbb{E}\left[ \left( \frac{\nabla \mu_u(y)}{\mu_u(y)} \right)^2 \middle| y \right] - 2\mathrm{A}_r(y) \mathbb{E}\left[ \left( \frac{\nabla \mu_u(y)}{\mu_u(y)} \right)^2 \mathrm{J}_\beta(j) \middle| y \right] \right].
$$

From Lemma 3 it can be seen that the task of finding the optimal baseline is in effect that of minimizing a quadratic for each observation $y \in \mathcal{Y}$. This gives the following theorem.

**Theorem 4** *For the controlled POMDP as in Lemma 3,*

$$\min_{A_r} \underline{\sigma}_r^2 = \sigma^2 - \mathbb{E}\left[\left(\mathbb{E}\left[\left(\frac{\nabla\mu_u(y)}{\mu_u(y)}\right)^2 \mathrm{J}_\beta(j)\,\middle|\, y\right]\right)^2 \middle/ \mathbb{E}\left[\left(\frac{\nabla\mu_u(y)}{\mu_u(y)}\right)^2 \middle|\, y\right]\right],$$

*and this minimum is attained with the baseline*

$$A_r^*(y) = \mathbb{E}\left[\left(\frac{\nabla\mu_u(y)}{\mu_u(y)}\right)^2 \mathrm{J}_\beta(j)\,\middle|\, y\right] \middle/ \mathbb{E}\left[\left(\frac{\nabla\mu_u(y)}{\mu_u(y)}\right)^2 \middle|\, y\right].$$

*Furthermore, the optimal* constant *baseline is*

$$A_r^* = \mathbb{E}\left[\left(\frac{\nabla\mu_u(y)}{\mu_u(y)}\right)^2 \mathrm{J}_\beta(j)\right] \middle/ \mathbb{E}\left(\frac{\nabla\mu_u(y)}{\mu_u(y)}\right)^2.$$

The following theorem shows that the variance of an estimate with an arbitrary baseline can be expressed as the sum of the variance with the optimal baseline and a certain squared weighted distance between the baseline function and the optimal baseline function.

**Theorem 5** *If* $A_r : \mathcal{Y} \to \mathbb{R}$ *is a baseline function,* $A_r^*$ *is the optimal baseline defined in Theorem 4, and* $\underline{\sigma}_{r*}^2$ *is the variance of the corresponding estimate, then*

$$\underline{\sigma}_r^2 = \underline{\sigma}_{r*}^2 + \mathbb{E}\left[\left(\frac{\nabla\mu_u(y)}{\mu_u(y)}\right)^2 (A_r(y) - A_r^*(y))^2\right],$$

*where* $i \sim \pi$, $y \sim \nu(i)$, *and* $u \sim \mu(y)$. *Furthermore, the same result is true for the case of* constant *baselines, with* $A_r(y)$ *replaced by an arbitrary constant baseline* $A_r$, *and* $A_r^*(y)$ *replaced by* $A_r^*$, *the optimum constant baseline defined in Theorem 4.*

For the constant baseline $A_r = \mathbb{E}_{i \sim \pi}[\mathrm{J}_\beta(i)]$, Theorem 5 implies that $\underline{\sigma}_r^2$ is equal to

$$\min_{A_r \in \mathbb{R}} \underline{\sigma}_r^2 + \left(\mathbb{E}\left(\frac{\nabla\mu_u(y)}{\mu_u(y)}\right)^2 \mathbb{E}[\mathrm{J}_\beta(j)] - \mathbb{E}\left[\left(\frac{\nabla\mu_u(y)}{\mu_u(y)}\right)^2 \mathrm{J}_\beta(j)\right]\right)^2 \middle/ \mathbb{E}\left(\frac{\nabla\mu_u(y)}{\mu_u(y)}\right)^2.$$

Thus, its performance depends on the random variables $(\nabla\mu_u(y)/\mu_u(y))^2$ and $\mathrm{J}_\beta(j)$; if they are nearly independent, $\mathbb{E}[\mathrm{J}_\beta]$ is a good choice.

## 3  Fixed Value Functions: Actor-Critic Methods

We consider an alteration of (1),

$$\tilde{\Delta}_T \stackrel{\mathrm{def}}{=} \frac{1}{T} \sum_{t=0}^{T-1} \frac{\nabla\mu_{U_t}(Y_t)}{\mu_{U_t}(Y_t)} \tilde{\mathrm{J}}_\beta(X_{t+1}), \tag{4}$$

for some fixed value function $\tilde{\mathrm{J}}_\beta : \mathcal{S} \to \mathbb{R}$. Define

$$A_\beta(j) \stackrel{\mathrm{def}}{=} \mathbb{E}\left[\sum_{k=0}^\infty \beta^k \mathrm{d}(X_k, X_{k+1})\,\middle|\, X_0 = j\right],$$

where $\mathrm{d}(i,j) \stackrel{\mathrm{def}}{=} \mathrm{r}(i) + \beta\tilde{\mathrm{J}}_\beta(j) - \tilde{\mathrm{J}}_\beta(i)$ is the temporal difference. Then it is easy to show that the estimate (4) has a bias of

$$\nabla_\beta \eta - \mathbb{E}\left[\tilde{\Delta}_T\right] = \mathbb{E}\left[\frac{\nabla\mu_u(y)}{\mu_u(y)} A_\beta(j)\right].$$

The following theorem gives a bound on the expected squared error of (4). The main tool in the proof is Theorem 1.

**Theorem 6** *Let $D = (\mathcal{S}, \mathcal{U}, \mathcal{Y}, P, \nu, \mathrm{r}, \mu)$ be a controlled POMDP satisfying Assumption 1. For a sample path from $D$, that is, $\{X_0 \sim \pi, Y_t \sim \nu(X_t), U_t \sim \mu(Y_t), X_{t+1} \sim P_{X_t}(U_t)\}$,*

$$\mathbb{E}\left[\left(\tilde{\Delta}_T - \nabla_\beta \eta\right)^2\right] \leq \frac{\Omega^*}{T} \mathrm{Var}_\pi\left(\frac{\nabla \mu_u(y)}{\mu_u(y)} \tilde{\mathrm{J}}_\beta(j)\right) + \left(\mathbb{E}\left[\frac{\nabla \mu_u(y)}{\mu_u(y)} \mathrm{A}_\beta(j)\right]\right)^2,$$

*where the second expectation is over $i \sim \pi$, $y \sim \nu(i)$, $u \sim \mu(y)$, and $j \sim P_i(u)$.*

If we write $\tilde{\mathrm{J}}_\beta(j) = \mathrm{J}_\beta(j) + v(j)$, then by selecting $v = (v(1), \ldots, v(|\mathcal{S}|))'$ from the right null space of the $K \times |\mathcal{S}|$ matrix $G$, where $G \overset{\text{def}}{=} \sum_{i,y,u} \pi_i \nu_y(i) \nabla \mu_u(y) P_i'(u)$, (4) will produce an unbiased estimate of $\nabla_\beta \eta$. An obvious example of such a $v$ is a constant vector, $(c, c, \ldots, c)' : c \in \mathbb{R}$. We can use this to construct a trivial example where (4) produces an unbiased estimate with zero variance. Indeed, let $D = (\mathcal{S}, \mathcal{U}, \mathcal{Y}, P, \nu, \mathrm{r}, \mu)$ be a controlled POMDP satisfying Assumption 1, with $\mathrm{r}(i) = c$, for some $0 < c \leq \mathbf{R}$. Then $\mathrm{J}_\beta(j) = c/(1-\beta)$ and $\nabla_\beta \eta = 0$. If we choose $v = (-c/(1-\beta), \ldots, -c/(1-\beta))'$ and $\tilde{\mathrm{J}}_\beta(j) = \mathrm{J}_\beta(j) + v(j)$, then $\frac{\nabla \mu_u(y)}{\mu_u(y)} \tilde{\mathrm{J}}_\beta(j) = 0$ for all $y, u, j$, and so (4) gives an unbiased estimate of $\nabla_\beta \eta$, with zero variance. Furthermore, for any $D$ for which there exists a pair $y, u$ such that $\mu_u(y) > 0$ and $\nabla \mu_u(y) \neq 0$, choosing $\tilde{\mathrm{J}}_\beta(j) = \mathrm{J}_\beta(j)$ gives a variance greater than zero—there is a non-zero probability event, $(X_t = i, Y_t = y, U_t = u, X_{t+1} = j)$, such that $\frac{\nabla \mu_u(y)}{\mu_u(y)} \mathrm{J}_\beta(j) \neq 0$.

# 4 Algorithms

Given a parameterized class of baseline functions $\{\mathrm{A}_r(\cdot, \theta) : \mathcal{Y} \to \mathbb{R} \,\big|\, \theta \in \mathbb{R}^L\}$, we can use Theorem 5 to bound the variance of our estimates. Computing the gradient of this bound with respect to the parameters $\theta$ of the baseline function allows a gradient optimization of the baseline. The GDORB Algorithm produces an estimate $\Delta_S$ of these gradients from a sample path of length $S$. Under the assumption that the baseline function and its gradients are uniformly bounded, we can show that these estimates converge to the gradient of $\underline{\sigma}_r^2$. We omit the details (see [8]).

**GDORB Algorithm:**

    **Given:** Controlled POMDP $(\mathcal{S}, \mathcal{U}, \mathcal{Y}, P, \nu, \mathrm{r}, \mu)$, parameterized baseline $\mathrm{A}_r$.
    set $z_0 = 0$ $(z_0 \in \mathbb{R}^L)$, $\Delta_0 = 0$ $(\Delta_0 \in \mathbb{R}^L)$
    **for all** $\{i_s, y_s, u_s, i_{s+1}, y_{s+1}\}$ generated by the POMDP **do**

$$z_{s+1} = \beta z_s + \nabla \mathrm{A}_r(y_s) \left(\frac{\nabla \mu_{u_s}(y_s)}{\mu_{u_s}(y_s)}\right)^2$$
$$\Delta_{s+1} = \Delta_s + \frac{1}{s+1}\left((\mathrm{A}_r(y_s) - \beta \mathrm{A}_r(y_{s+1}) - \mathrm{r}(x_{s+1})) z_{s+1} - \Delta_s\right)$$

    **end for**

For a parameterized class of fixed value functions $\{\tilde{\mathrm{J}}_\beta(\cdot, \theta) : \mathcal{S} \to \mathbb{R} \,\big|\, \theta \in \mathbb{R}^L\}$, we can use Theorem 6 to bound the expected squared error of our estimates, and compute the gradient of this bound with respect to the parameters $\theta$ of the baseline function. The GBTE Algorithm produces an estimate $\Delta_S$ of these gradients from a sample path of length $S$. Under the assumption that the value function and its gradients are uniformly bounded, we can show that these estimates converge to the true gradient.

**GBTE Algorithm:**

    **Given:** Controlled POMDP $(\mathcal{S}, \mathcal{U}, \mathcal{Y}, P, \nu, \mathrm{r}, \mu)$, parameterized fixed value function $\tilde{\mathrm{J}}_\beta$.
    set $z_0 = 0$ $(z_0 \in \mathbb{R}^K)$, $\Delta A_0 = 0$ $(\Delta A_0 \in \mathbb{R}^{1 \times L})$, $\Delta B_0 = 0$ $(\Delta B_0 \in \mathbb{R}^K)$, $\Delta C_0 = 0$ $(\Delta C_0 \in \mathbb{R}^K)$ and $\Delta D_0 = 0$ $(\Delta D_0 \in \mathbb{R}^{K \times L})$

**for all** $\{i_s, y_s, u_s, i_{s+1}, i_{s+2}\}$ generated by the POMDP **do**

$$z_{s+1} = \beta z_s + \frac{\nabla \mu_{u_s}(y_s)}{\mu_{u_s}(y_s)}$$

$$\Delta A_{s+1} = \Delta A_s + \tfrac{1}{s+1}\left(\left(\frac{\nabla \mu_{u_s}(y_s)}{\mu_{u_s}(y_s)}\tilde{\mathrm{J}}_\beta(i_{s+1})\right)'\left(\frac{\nabla \mu_{u_s}(y_s)}{\mu_{u_s}(y_s)}\left(\nabla\tilde{\mathrm{J}}_\beta(i_{s+1})\right)'\right) - \Delta A_s\right)$$

$$\Delta B_{s+1} = \Delta B_s + \tfrac{1}{s+1}\left(\frac{\nabla \mu_{u_s}(y_s)}{\mu_{u_s}(y_s)}\tilde{\mathrm{J}}_\beta(i_{s+1}) - \Delta B_s\right)$$

$$\Delta C_{s+1} = \Delta C_s + \tfrac{1}{s+1}\left(\left(\mathrm{r}(i_{s+1}) + \beta\tilde{\mathrm{J}}_\beta(i_{s+2}) - \tilde{\mathrm{J}}_\beta(i_{s+1})\right)z_{s+1} - \Delta C_s\right)$$

$$\Delta D_{s+1} = \Delta D_s + \tfrac{1}{s+1}\left(\frac{\nabla \mu_{u_s}(y_s)}{\mu_{u_s}(y_s)}\left(\nabla\tilde{\mathrm{J}}_\beta(i_{s+1})\right)' - \Delta D_s\right)$$

$$\Delta_{s+1} = \left(\tfrac{\Omega^*}{T}\Delta A_{s+1} - \tfrac{\Omega^*}{T}\Delta B'_{s+1}\Delta D_{s+1} - \Delta C'_{s+1}\Delta D_{s+1}\right)'$$

**end for**

## 5   Experiments

Experimental results comparing these GPOMDP variants for a simple three state MDP (described in [5]) are shown in Figure 1. The exact value function plots show how different choices of baseline and fixed value function compare when all algorithms have access to the exact value function $\mathrm{J}_\beta$. Using the expected value function as a baseline was an improvement over GPOMDP. Using the optimum, or optimum constant, baseline was a further improvement, each performing comparably to the other. Using the pre-trained fixed value function was also an improvement over GPOMDP, showing that selecting the true value function was indeed not the best choice in this case. The trained fixed value function was not optimal though, as $\mathrm{J}_\beta(j) - A_r^*$ is a valid choice of fixed value function.

The optimum baseline, and fixed value function, will not normally be known. The online plots show experiments where the baseline and fixed value function were trained using online gradient descent whilst the performance gradient was being estimated, using the same data. Clear improvement over GPOMDP is seen for the online trained baseline variant. For the online trained fixed value function, improvement is seen until $T$ becomes—given the simplicity of the system—very large.

## References

[1] L. Baird and A. Moore. Gradient descent for general reinforcement learning. In *Advances in Neural Information Processing Systems 11*, pages 968–974. MIT Press, 1999.

[2] P. L. Bartlett and J. Baxter. Estimation and approximation bounds for gradient-based reinforcement learning. *Journal of Computer and Systems Sciences*, 2002. To appear.

[3] A. G. Barto, R. S. Sutton, and C. W. Anderson. Neuronlike adaptive elements that can solve difficult learning control problems. *IEEE Transactions on Systems, Man, and Cybernetics*, SMC-13:834–846, 1983.

[4] J. Baxter and P. L. Bartlett. Infinite-horizon gradient-based policy search. *Journal of Artificial Intelligence Research*, 15:319–350, 2001.

[5] J. Baxter, P. L. Bartlett, and L. Weaver. Infinite-horizon gradient-based policy search: II. Gradient ascent algorithms and experiments. *Journal of Artificial Intelligence Research*, 15:351–381, 2001.

[6] M. Evans and T. Swartz. *Approximating integrals via Monte Carlo and deterministic methods*. Oxford University Press, 2000.

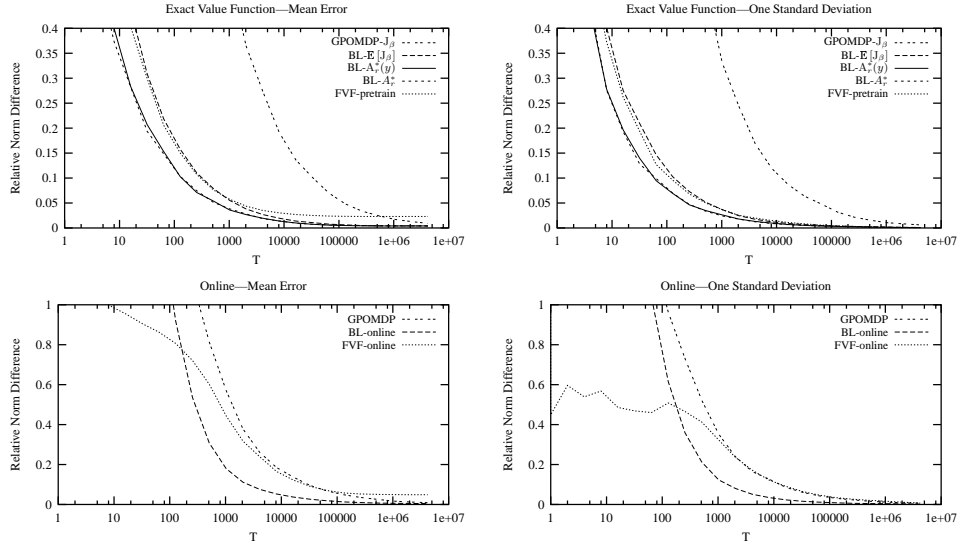

Figure 1: Three state experiments—relative norm error $\|\Delta_{\text{est}} - \nabla\eta\| / \|\nabla\eta\|$. Exact value function plots compare mean error and standard deviations for gradient estimates (with knowledge of $J_\beta$) computed by: GPOMDP [GPOMDP-$J_\beta$]; with baseline $A_r = \mathbb{E}[J_\beta]$ [BL-$\mathbb{E}[J_\beta]$]; with optimum baseline [BL-$A_r^*(y)$]; with optimum constant baseline [BL-$A_r^*$]; with pre-trained fixed value function [FVF-pretrain]. Online plots do a similar comparison of estimates computed by: GPOMDP [GPOMDP]; with online trained baseline [BL-online]; with online trained fixed value function [FVF-online]. The plots were computed over 500 runs (1000 for FVF-online), with $\beta = 0.95$. $\Omega^*/T$ was set to 0.001 for FVF-pretrain, and 0.01 for FVF-online.

[7] P. W. Glynn. Likelihood ratio gradient estimation for stochastic systems. *Communications of the ACM*, 33:75–84, 1990.

[8] E. Greensmith, P. L. Bartlett, and J. Baxter. Variance reduction techniques for gradient estimates in reinforcement learning. Technical report, ANU, 2002.

[9] H. Kimura, K. Miyazaki, and S. Kobayashi. Reinforcement learning in POMDPs with function approximation. In D. H. Fisher, editor, *Proceedings of the Fourteenth International Conference on Machine Learning (ICML'97)*, pages 152–160, 1997.

[10] V. R. Konda and J. N. Tsitsiklis. Actor-Critic Algorithms. In *Advances in Neural Information Processing Systems 12*, pages 1008–1014. MIT Press, 2000.

[11] P. Marbach and J. N. Tsitsiklis. Simulation-Based Optimization of Markov Reward Processes. Technical report, MIT, 1998.

[12] R. Y. Rubinstein. How to optimize complex stochastic systems from a single sample path by the score function method. *Ann. Oper. Res.*, 27:175–211, 1991.

[13] R. S. Sutton and A. G. Barto. *Reinforcement Learning: An Introduction*. MIT Press, Cambridge MA, 1998. ISBN 0-262-19398-1.

[14] R. S. Sutton, D. McAllester, S. Singh, and Y. Mansour. Policy Gradient Methods for Reinforcement Learning with Function Approximation. In *Advances in Neural Information Processing Systems 12*, pages 1057–1063. MIT Press, 2000.

[15] R. J. Williams. Simple Statistical Gradient-Following Algorithms for Connectionist Reinforcement Learning. *Machine Learning*, 8:229–256, 1992.
